# Statistical Tests for Optimization Efficiency

**Levi Boyles, Anoop Korattikara, Deva Ramanan, Max Welling**
Department of Computer Science
University of California, Irvine
Irvine, CA 92697-3425
{lboyles},{akoratti},{dramanan},{welling}@ics.uci.edu

## Abstract

Learning problems, such as logistic regression, are typically formulated as pure optimization problems defined on some loss function. We argue that this view ignores the fact that the loss function depends on stochastically generated data which in turn determines an intrinsic scale of precision for statistical estimation. By considering the statistical properties of the update variables used during the optimization (e.g. gradients), we can construct frequentist hypothesis tests to determine the reliability of these updates. We utilize subsets of the data for computing updates, and use the hypothesis tests for determining when the batch-size needs to be increased. This provides computational benefits and avoids overfitting by stopping when the batch-size has become equal to size of the full dataset. Moreover, the proposed algorithms depend on a single interpretable parameter – the probability for an update to be in the wrong direction – which is set to a single value across all algorithms and datasets. In this paper, we illustrate these ideas on three $L_1$ regularized coordinate descent algorithms: $L_1$-regularized $L_2$-loss SVMs, $L_1$-regularized logistic regression, and the Lasso, but we emphasize that the underlying methods are much more generally applicable.

## 1 Introduction

There is an increasing tendency to consider machine learning as a problem in optimization: define a loss function, add constraints and/or regularizers and formulate it as a preferably convex program. Then, solve this program using some of the impressive tools from the optimization literature. The main purpose of this paper is to point out that this "reduction to optimization" ignores certain important statistical features that are unique to statistical estimation. The most important feature we will exploit is the fact that the *statistical properties of an estimation problem determine an intrinsic scale of precision* (that is usually much larger than machine precision). This implies immediately that optimizing parameter-values beyond that scale is pointless and may even have an adverse affect on generalization when the underlying model is complex. Besides a natural stopping criterion it also leads to much faster optimization before we reach that scale by realizing that far away from optimality we need much less precision to determine a parameter update than when close to optimality. These observations can be incorporated in many off-the-shelves optimizers and are often orthogonal to speed-up tricks in the optimization toolbox.

The intricate relationship between computation and estimation has been pointed out before in [1] and [2] where asymptotic learning rates were provided. One of the important conclusions was that a not so impressive optimization algorithm such as stochastic gradient descent (SGD) can be nevertheless a very good learning algorithm because it can process more data per unit time. Also in [3] (sec. 5.5) the intimate relationship between computation and model fitting is pointed out. [4] gives bounds on the generalization risk for online algorithms, and [5] shows how additional data can be used to reduce running time for a fixed target generalization error. Regret-minimizing algorithms ([6], [7]) are another way to account for the interplay between learning and computation. Hypothesis testing has been exploited for computational gains before in [8].

Our method exploits the fact that loss functions are random variables subject to uncertainty. In a frequentist world we may ask how different the value of the loss would have been if we would have sampled another dataset of the same size from a single shared underlying distribution. The role of an optimization algorithm is then to propose parameter updates that will be accepted or rejected on statistical grounds. The test we propose determines whether the direction of a parameter update is correct with high probability. If we do not pass our tests when using all the available data-cases then we stop learning (or alternatively we switch to sampling or bagging), because we have reached the intrinsic scale of precision set by the statistical properties of the estimation problem.

However, we can use the same tests to speed up the optimization process itself, that is before we reach the above stopping criterion. To see that, imagine one is faced with an infinite dataset. In batch mode, using the whole (infinite) dataset, one would not take a single optimization step in finite time. Thus, one should really be concerned with making as much progress as possible *per computational unit*. Hence, one should only use a subset of the total available dataset. Importantly, the optimal batch-size depends on where we are in the learning process: far away from convergence we only need a rough idea of where to move which requires very few data-cases. On the other hand, the closer we get to the true parameter value, the more resolution we need. Thus, the computationally optimal batch-size is a function of the residual estimation error. Our algorithm adaptively grows a subset of the data by requiring that we have just enough precision to confidently move in the correct direction. Again, when we have exhausted all our data we stop learning.

Our algorithm heavily relies on the central limit tendencies of large sums of random variables. Fortunately, many optimization algorithms are based on averages over data-cases. For instance, gradient descent falls in this class, as the gradient is defined by an average (or sum). As in [11], with large enough batch sizes we can use the Central Limit Theorem to claim that the average gradients are normally distributed and estimate their variance without actually seeing more data (this assumption is empirically verified in section 5.2). We have furthermore implemented methods to avoid testing updates for parameters which are likely to fail their test. This ensures that we approximately visit the features with their correct frequency (i.e. important features may require more updates than unimportant ones).

In summary, the main contribution of this paper is to introduce a class of algorithms with the following properties.

- They depend on a single interpretable parameter $\epsilon$ – the probability to update parameters in the wrong direction. Moreover, the performance of the algorithms is relatively insensitive to the exact value we choose.
- They have a natural, inbuilt stopping criterion. The algorithms terminate when the probability to update the parameters in the wrong direction can not be made smaller than $\epsilon$.
- They are applicable to wide range of loss functions. The only requirement is that the updates depend on sums of random variables.
- They inherit the convergence guarantees of the optimization method under consideration. This follows because the algorithms will eventually consider all the data.
- They achieve very significant speedups in learning models from data. Throughout the learning process they determine the size of the data subset required to perform updates that move in the correct direction with probability at least $1 - \epsilon$.

We emphasize that our framework is generally applicable. In this paper we show how these considerations can be applied to $L_1$-regularized coordinate descent algorithms: $L_1$-regularized $L_2$-loss SVMs, $L_1$-regularized logistic regression, and Lasso [9]. Coordinate descent algorithms are convenient because they do not require any tuning of hyper-parameters to be effective, and are still efficient when training sparse models. Our methodology extends these algorithms to be competitive for dense models and for $N >> p$. In section 2 we review the coordinate descent algorithms. Then, in section 3.2 we formulate our hypothesis testing framework, followed by a heuristic for predicting hypothesis test failures in section 4. We report experimental results in section 5 and we end with conclusions.

## 2   Coordinate Descent

We consider L1-regularized learning problems where the loss is defined as a statistical average over $N$ datapoints:

$$f(\beta) = \gamma ||\beta||_1 + \frac{1}{2N} \sum_{i=1}^{N} \text{loss}(\beta, x_i, y_i) \quad \text{where} \quad \beta, x_i \in \mathbb{R}^p \tag{1}$$

We will consider continously-differentiable loss functions (squared hinge-loss, log-loss, and squared-loss) that allow for the use of efficient coordinate-descent optimization algorithms, where

each parameter is updated $\beta_j^{new} \leftarrow \beta_j + d_j$ with:

$$d_j = \underset{d}{\text{argmin}}\, f(\beta + de_j) \qquad f(\beta + de_j) = |\beta_j + d| + L_j(d; \beta) + \text{const} \qquad (2)$$

where $L_j(d; \beta) = \frac{1}{2N} \sum_{i=1}^{N} \text{loss}(\beta + de_j, x_i, y_i)$ and $e_j$ is the $j^{th}$ standard basis vector. To solve the above, we perform a second-order Taylor expansion of the partial loss $L_j(d; \beta)$:

$$f(\beta + de_j) \approx |\beta_j + d| + L_j'(0; \beta)d + \frac{1}{2}L_j''(0; \beta)d^2 + \text{const} \qquad (3)$$

[10] show that the minimum of the approximate objective (3) is obtained with:

$$d_j = \begin{cases} -\frac{L_j'(0,\beta)+\gamma}{L_j''(0,\beta)} & \text{if } L_j'(0, \beta) + \gamma \leq L_j''(0, \beta)\beta_j \\ -\frac{L_j'(0,\beta)-\gamma}{L_j''(0,\beta)} & \text{if } L_j'(0, \beta) - \gamma \geq L_j''(0, \beta)\beta_j \\ -\beta_j & \text{otherwise} \end{cases} \qquad (4)$$

For quadratic loss functions, the approximation in (3) is exact. For general convex loss functions, one can optimize (2) by repeatedly linearizing and applying the above update. We perform a single update per parameter during the cyclic iteration over parameters. Notably, the partial derivatives are functions of statistical averages computed over $N$ training points. We show that one can use frequentist hypothesis tests to elegantly manage the amount of data needed ($N$) to reliably compute these quantities.

## 2.1 $L_1$-regularized $L_2$-loss SVM

Using a squared hinge-loss function in (1), we obtain a $L_1$-regularized $L_2$-loss SVM:

$$\text{loss}_{SVM} = \max(0, 1 - y_i \beta^T x_i)^2 \qquad (5)$$

Appendix F of [10] derive the corresponding partial derivatives, where the second-order statistic is approximate because the squared hinge-loss is not twice differentiable:

$$L_j'(0, \beta) = -\frac{1}{N} \sum_{i \in I(\beta)} y_i x_{ij} b_i(\beta) \qquad L_j''(0, \beta) = \frac{1}{N} \sum_{i \in I(\beta)} x_{ij}^2 \qquad (6)$$

where $b_i(\beta) = 1 - y_i \beta^T x_i$ and $I(\beta) = \{i | b_i(\beta) > 0\}$. We write $x_{ij}$ for the $j^{th}$ element of datapoint $x_i$. In [10], each parameter is updated until convergence, using a line-search for each update, whereas we simply check that $L''$ term is not ill formed rather than performing a line search.

## 2.2 $L_1$-regularized Logistic Regression

Using a log-loss function in (1), we obtain a $L_1$-regularized logistic regression model:

$$\text{loss}_{log} = \log(1 + e^{-y_i \beta^T x_i}) \qquad (7)$$

Appendix G of [10] derive the corresponding partial derivatives:

$$L_j'(0, \beta) = \frac{1}{2N} \sum_{i=1}^{N} \frac{-x_{ij}}{1 + e^{y_i \beta^T x_i}} \qquad L_j''(0, \beta) = \frac{1}{2N} \sum_{i=1}^{N} \left(\frac{x_{ij}}{1 + e^{y_i \beta^T x_i}}\right)^2 e^{y_i \beta^T x_i} \qquad (8)$$

## 2.3 $L_1$-regularized Linear Regression (Lasso)

Using a quadratic loss function in (1), we obtain a $L_1$-regularized linear regression, or LASSO, model:

$$\text{loss}_{quad} = (y_i - \beta^T x_i)^2 \qquad (9)$$

The corresponding partial derivatives [9] are:

$$L_j'(0, \beta) = -\frac{1}{N} \sum_{i=1}^{N} (y_i - \beta^T x_i)x_{ij} \qquad L_j''(0, \beta) = \frac{1}{N} \sum_{i=1}^{N} x_{ij} x_{ij} \qquad (10)$$

Because the Taylor expansion is exact for quadratic loss functions, we can directly write the closed form solution for parameter $\beta_j^{new} = S(\alpha_j, \gamma)$ where

$$\alpha_j = \frac{1}{N} \sum_{i=1}^{N} x_{ij}(y_i - \tilde{y}_i^{(j)}) \qquad S(\alpha, \gamma) = \begin{cases} \alpha - \gamma & \alpha > 0, \gamma < |\alpha| \\ \alpha + \gamma & \alpha < 0, \gamma < |\alpha| \\ 0 & \gamma \geq |\alpha| \end{cases} \tag{11}$$

where $\tilde{y}_i^{(j)} = \sum_{k \neq j} x_{ik}\beta_k$ is the prediction made with all parameters except $\beta_j$ and $S$ is a "soft-threshold" function that is zero for an interval of $2\gamma$ about the origin, and shrinks the magnitude of the input $\alpha$ by $\gamma$ outside of this interval. We can use this expression as an estimator for $\beta$ from a dataset $\{x_i, y_i\}$. The above update rule assumes standardized data ($\frac{1}{N} \sum_i x_{ij} = 0$, $\frac{1}{N} \sum_i x_{ij}^2 = 1$), but it is straightforward to extend for the general case.

## 3 Hypothesis Testing

Each update $\beta_j^{new} = \beta_j + d_j$ is computed using a statistical average over a batch of $N$ training points. We wish to estimate the reliability of an update as a function of $N$. To do so, we model the current $\beta$ vector as a *fixed constant* and the $N$ training points as *random variables* drawn from an underlying joint density $p(x, y)$. This also makes the proposed updates $d_j$ and $\beta_j^{new}$ random variables because they are functions of the training points. In the following we will make an explicit distinction between random variables, e.g. $\beta_j^{new}, d_j, x_{ij}, y_i$ and their instantiations, $\hat{\beta}_j^{new}, \hat{d}_j, \hat{x}_{ij}, \hat{y}_i$. We would like to determine whether or not a particular update is statistically justified. To this end, we use hypothesis tests where if there is high uncertainty in the direction of the update, we say this update is not justified and the update is not performed. For example, if our proposed update $\hat{\beta}_j^{new}$ is positive, we want to ensure that $P(\beta_j^{new} < 0)$ is small.

### 3.1 Algorithm Overview

We propose a "growing batch" algorithm for handling very large or infinite datasets: first we select a very small subsample of the data of size $N_b \ll N$, and optimize until the entire set of parameters are failing their hypothesis tests (described in more detail below). We then query more data points and include them in our batch, reducing the variance of our estimates and making it more likely that they will pass their tests. We continue adding data to our batch until we are using the full dataset of size $N$. Once all of the parameters are failing their hypothesis tests on the full batch of data, we stop training. The reasoning behind this is, as argued in the introduction, that at this point we do not have enough evidence for even determining the direction in which to update, which implies that further optimization would result in overfitting. Thus, our algorithm behaves like a stochastic online algorithm during early stages and like a batch algorithm during later stages, equipped with a natural stopping condition.

In our experiments, we increase the batch size $N_b$ by a factor of 10 once all parameters fail their hypothesis tests for a given batch. Values in the range 2-100 also worked well, however, we chose 10 as it works very well for our implementation.

### 3.2 Lasso

For quadratic loss functions with standardized variables, we can directly analyze the densities of $d_j, \beta_j^{new}$. We accept an update if the sign of $d_j$ can be estimated with sufficient probability. Central to our analysis is $\alpha_j$ (11), which is equivalent to $\beta_j^{new}$ for the unregularized case $\gamma = 0$. We rewrite it as:

$$\alpha_j = \frac{1}{N} \sum_{i=1}^{N} z_{ij}(\beta) \quad \text{where} \quad z_{ij}(\beta) = x_{ij}(y_i - \tilde{y}_i^{(j)}) \tag{12}$$

Because $z_{ij}(\beta)$ are given by a fixed transformation of the iid training points, they themselves are iid. As $N \to \infty$, we can appeal to the Central Limit Theorem and model $\alpha_j$ as distributed as a standard Normal: $\alpha_j \sim \mathcal{N}(\mu_{\alpha_j}, \sigma_{\alpha_j})$, where $\mu_{\alpha_j} = E[z_{ij}]$, $\forall i$ and $\sigma_{\alpha_j}^2 = \frac{1}{N} Var(z_{ij})$ $\forall i$. Empirical justification of normality of these quantities is given in section 5.2. So, for any given $\alpha_j$, we can provide estimates

$$E[z_{ij}] \approx \overline{\hat{z}_j} = \frac{1}{N} \sum_i \hat{z}_{ij} \qquad Var(z_{ij}) \approx \sigma_{\hat{z}_j}^2 = \frac{1}{N-1} \sum_i (\hat{z}_{ij} - \overline{\hat{z}_j})^2 \tag{13}$$

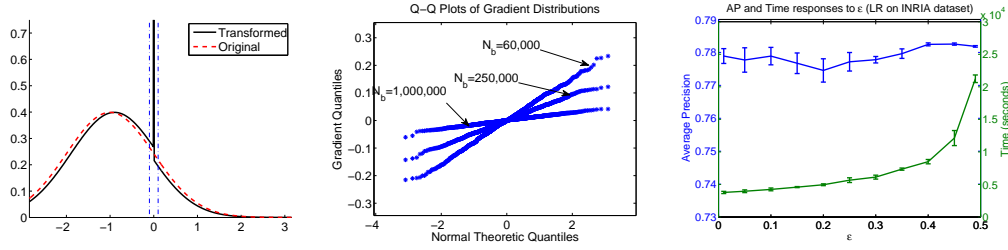

Figure 1: (**left**) A Gaussian distribution and the distribution resulting from applying the transformation $S$, with $\gamma = .1$. The interval that is "squashed" is shown by the dash-dotted blue lines. (**middle**) Q-Q plot demonstrating the normality of the gradients on the $L_1$-regularized $L_2$-loss SVM, computed at various stages of the algorithm (i.e. at different batch-sizes $N_b$ and models $\beta$). Straight lines provide evidence that the empirical distribution is close to normality. (**right**) Plot showing the behavior of our algorithm with respect to $\epsilon$, using logistic regression on the INRIA dataset. $\epsilon = 0$ corresponds to an algorithm which never updates, and $\epsilon = 0.5$ corresponds to an algorithm which always updates (with no stopping criteria), so for these experiments $\epsilon$ was chosen in the range $[.01, .49]$. Error bars denote a single standard deviation.

which in turn provide estimates for $\mu_{\alpha_j}$ and $\sigma_{\alpha_j}$. We next apply the soft threshold function $S$ to $\alpha_j$ to obtain $\beta_j^{new}$, a random variable whose pdf is a Gaussian which has a section of width $2\gamma$ "squashed" to zero into a single point of probability mass, with the remaining density shifted towards zero by a magnitude $\gamma$. This is illustrated in Figure 1. Our criterion for accepting an update is that it moves towards the true solution with high probability. Let $\hat{d}_j$ be the realization of the random variable $d_j = \beta_j^{new} - \beta_j$, computed from the sample batch of $N$ training points. If $\hat{d}_j > 0$, then we want $P(d_j \leq 0)$ to be small, and vice versa. Specifically, for $\hat{d}_j > 0$, we want $P(d_j \leq 0) < \epsilon$, where

$$P(d_j \leq 0) = P(\beta_j^{new} \leq \beta_j) = \begin{cases} \Phi\left(\frac{\beta_j - (\mu_{\alpha_j} + \gamma)}{\sigma_{\alpha_j}}\right) & \text{if } \beta_j < 0 \\ \Phi\left(\frac{\beta_j - (\mu_{\alpha_j} - \gamma)}{\sigma_{\alpha_j}}\right) & \text{if } \beta_j \geq 0 \end{cases} \tag{14}$$

where $\Phi(\cdot)$ denotes the cdf for the standard Normal. This distribution can be derived from its two underlying Gaussians, one with mean $\mu_{\alpha_j} + \gamma$ and one with mean $\mu_{\alpha_j} - \gamma$. Similarly, one can define an analgous test of $P(d_j \geq 0) < \epsilon$ for $\hat{d}_j < 0$. These are the hypothesis test equations for a single coordinate, so this test is performed once for each coordinate at its iteration in the coordinate descent algorithm. If a coordinate update fails its test, then we assume that we do not have enough evidence to perform an update on the coordinate, and do not update. Note that, since we are potentially rejecting many updates, significant computation could be going to "waste," as we are computing updates without using them. Methods to address this follow in section 4.

### 3.3 Gradient-Based Hypothesis Tests

For general convex loss functions, it is difficult to construct a pdf for $d_j$ and $\beta_j^{new}$. Instead, we accept an update $\beta_j^{new}$ if the sign of the partial derivative $\frac{\partial f(\beta)}{\partial \beta_j}$ can be estimated with sufficient reliability. Because $f(\beta)$ may be nondifferentiable, we define $\partial_j f(\beta)$ to be the set of 1D subgradients, or lower tangent planes, at $\beta$ along direction $j$. The minimal (in magnitude) subgradient $g_j$, associated with the flattest lower tangent, is:

$$g_j = \begin{cases} \alpha_j - \gamma & \text{if } \beta_j < 0 \\ \alpha_j + \gamma & \text{if } \beta_j > 0 \\ S(\alpha_j, \gamma) & \text{otherwise} \end{cases} \quad \text{where} \quad \alpha_j = L_j'(0, \beta) = \frac{1}{N} \sum_{i=1}^{N} z_{ij} \tag{15}$$

where $z_{ij}(\beta) = -2y_i x_{ij} b_i(\beta)$ for the squared hinge-loss and $z_{ij}(\beta) = \frac{x_{ij}}{1 + e^{y_i \beta^T x_i}}$ for log-loss. Appealing to the same arguements as in Sec.3.2, one can show that $\alpha_j \sim \mathcal{N}(\mu_{\alpha_j}, \sigma_{\alpha_j})$ where $\mu_{\alpha_j} = E[z_{ij}], \forall i$ and $\sigma_{\alpha_j}^2 = \frac{1}{N} Var(z_{ij}) \forall i$. Thus the pdf of subgradient $g$ is a Normal shifted by $\gamma \text{sign}(\beta_j)$ in the case where $\beta_j \neq 0$, or a Normal transformed by the function $S(\alpha_j, \gamma)$ in the case $\beta_j = 0$.

To formulate our hypothesis test, we write $\hat{g}_j$ as the realization of random variable $g_j$, computed from the batch of $N$ training points. We want to take an update only if our update is in the correct

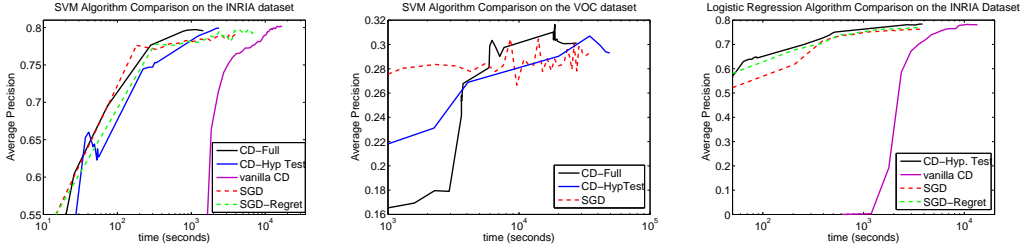

Figure 2: Plot comparing various algorithms for the $L_1$-regularized $L_2$-loss SVM on the INRIA dataset (**left**) and the VOC dataset (**middle**), and for the $L_1$-regularized logistic-regression on INRIA (**right**) using $\epsilon = 0.05$. "CD-Full" denotes our method using all applicable heuristic speedups, "CD-Hyp Testing" does not use the shrinking heuristic while "vanilla CD" simply performs coordinate descent without any speedup methods. "SGD" is stochastic gradient descent with an annealing schedule. Optimization of the hyper-parameters of the annealing schedule (on train data) was not included in the total runtime. Note that our method achieves the optimal precision faster than SGD and also stops learning approximately when overfitting sets in.

direction with high probability: for $\hat{g}_j > 0$, we want $P(g_j \leq 0) < \epsilon$, where

$$P(g_j \leq 0) = \begin{cases} \Phi\left(\frac{0-(\mu_{\alpha_j}-\gamma)}{\sigma_{\alpha_j}}\right) & \text{if } \beta_j \leq 0 \\ \Phi\left(\frac{0-(\mu_{\alpha_j}+\gamma)}{\sigma_{\alpha_j}}\right) & \text{if } \beta_j > 0 \end{cases} \tag{16}$$

We can likewise define a test of $P(g_j \geq 0) < \epsilon$ which we use to accept updates given a negative estimated gradient $\hat{g}_j < 0$.

## 4   Additional Speedups

It often occurs that many coordinates will fail their respective hypothesis tests for several consecutive iterations, so predicting these consecutive failures and skipping computations on these coordinates could potentially save computation. We employ a simple heuristic towards these matters based on a few observations (where for simplified notation we drop the subscript $j$):

1. If the set of parameters that are updating remains constant between updates, then for a particular coordinate, the change in the gradient from one iteration to the next is roughly constant. This is an empirical observation.
2. When close to the solution, $\sigma_\alpha$ remains roughly constant.

We employ a heuristic which is a complicated instance of a simple idea: if the value $a(0)$ of a variable of interest is changing at a constant rate $r$, we can predict its value at time $t$ with $a(t) = a(0) + rt$. In our case, we wish to predict when the gradient will have moved to a point where the associated hypothesis test will pass.

First, we will consider the unregularized case ($\gamma = 0$), wherein $g = \alpha$. We wish to detect when the gradient will result in the hypothesis test passing, that is, we want to find the values $\mu_\alpha \approx \hat{\alpha}$, where $\hat{\alpha}$ is a realization of the random variable $\alpha$, such that $P(g \geq 0) = \epsilon$ or $P(g \leq 0) = \epsilon$. For this purpose, we need to draw the distinction between an update which was taken, and one which is proposed but for which the hypothesis test failed. Let the set of accepted updates be indexed by $t$, as in $\hat{g}_t$, and let the set of potential updates, after an accepted update at time $t$, be indexed by $s$, as in $\hat{g}_t(s)$. Thus the algorithm described in the previous section will compute $\hat{g}_t(1)...\hat{g}_t(s^*)$ until the hypothesis test passes for $\hat{g}_t(s^*)$, and we then set $\hat{g}_{t+1}(0) = \hat{g}_t(s^*)$, and perform an update to $\beta$ using $\hat{g}_{t+1}(0)$. Ideally, we would prefer not to compute $\hat{g}_t(1)...\hat{g}_t(s^*-1)$ at all, and instead only compute the gradient when we know the hypothesis test will pass, $s^*$ iterations after the last accept.

Given that we have some scheme from skipping $k$ iterations, we estimate a "velocity" at which $\hat{g}_t(s) = \hat{\alpha}_t(s)$ changes: $\Delta_e \equiv \frac{\hat{\alpha}_t(s)-\hat{\alpha}_t(s-k-1)}{k+1}$. If, for instance, $\Delta_\alpha > 0$, we can compute the value of $\hat{\alpha}$ at which the hypothesis test will pass, assuming $\sigma_\alpha$ remains constant, by setting $P(g \leq 0|\mu_\alpha = \alpha_{pass}) = \epsilon$, and subsequently we can approximate the number of iterations to skip next[1]:

$$\alpha_{pass} = -\sigma_\alpha \Phi^{-1}(\epsilon) \qquad k_{skip} \leftarrow \frac{\alpha_{pass} - \hat{\alpha}_t(s)}{\Delta_\alpha} \tag{17}$$

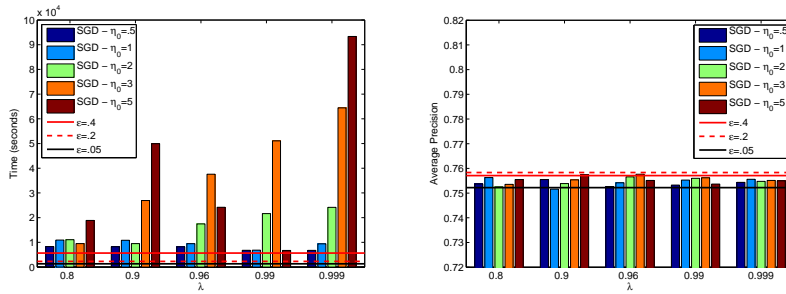

Figure 3: Comparison of our Lasso algorithm against SGD across various hyper-parameter settings for the exponential annealing schedule. Our algorithm is marked by the horizontal lines, with $\epsilon \in \{0.05, 0.2, 0.4\}$. Note that all algorithms have very similar precision scores in the interval $[0.75 - 0.76]$. For values of $\lambda = \{0.8, 0.9, 0.96, 0.99, 0.999\}$, SGD gives a good score, however, picking $\eta_0 > 1$ had an adverse effect on the optimization speed. Our method converged faster then SGD with the best annealing schedule.

The regularized case with $\beta_j > 0$ is equivalent to the unregularized case where $g = \alpha + \gamma$, and we solve for the value of $\alpha$ that will allow the test to pass via $P(g \leq 0 | \mu_\alpha = \alpha_{pass}) = \epsilon$:

$$\alpha_{pass} = -\sigma_\alpha \Phi^{-1}(\epsilon) - \gamma \qquad (18)$$

Similarly, the case with $\beta_j \leq 0$ is equivalent to the unregularized case where $g = \alpha - \gamma$: $\alpha_{pass} = -\sigma_\alpha \Phi^{-1}(\epsilon) + \gamma$. For the case where $\Delta_\alpha < 0$, we solve for $P(g \geq 0 | \mu_\alpha = \alpha_{pass}) = \epsilon$. This gives $\alpha_{pass} = -\sigma_\alpha \Phi^{-1}(1 - \epsilon) + \gamma$ if $\beta_j < 0$ and $\alpha_{pass} = -\sigma_\alpha \Phi^{-1}(1 - \epsilon) - \gamma$ otherwise. A similar heuristic for the Lasso case can also be derived.

### 4.1 Shrinking Strategy

It is common in SVM algorithms to employ a "shrinking" strategy in which datapoints which do not contribute to the loss are removed from future computations. Specifically, if a data point $(x_i, y_i)$ has the property that $b_i = 1 - y_i \beta^T x_i < \epsilon_{shrink} < 0$, for some $\epsilon_{shrink}$, then the data point is removed from the current batch. Data points removed from earlier batches in the optimization are still candidates for future batches. We employ this heuristic in our SVM implementation, and Figure 2 shows the relative performance between including this heuristic and not.

## 5 Experiments

### 5.1 Datasets

We provide experimental results for the task of visual object detection, building on recent successful approaches that learn linear scanning-window classifiers defined on Histograms of Oriented Gradients (HOG) descriptors [12, 13]. We train and evaluate a pedestrain detector using the INRIA dataset [12], where $(N, p) = (5e6, 1100)$. We also train and evaluate a car detector using the 2007 PASCAL VOC dataset [13], where $(N, p) = (6e7, 1400)$. For both datasets, we measure performance using the standard PASCAL evaluation protocol of average precision (with 50% overlap of predicted/ground truth bounding boxes). On such large training sets, one would expect delicately-tuned stochastic online algorithms (such as SGD) to outperform standard batch optimization (such as coordinate descent). We show that our algorithm exhibits the speed of the former with the reliability of the latter.

### 5.2 Normality Tests

In this section we empirically verify the normality claims on the INRIA dataset. Because the negative examples in this data are comprised of many overlapping windows from images, we may expect this non-iid property to damage any normality properties of our updates. For these experiments, we focus on the gradients of the $L_1$-regularized, $L_2$-loss SVM computed during various stages of the optimization process. Figure 1 shows quantile-quantile plots of the average gradient, computed over different subsamples of the data of fixed size $N_b$, versus the standard Normal. Experiments for smaller $N$ ($\approx 100$) and random $\beta$ give similar curves. We conclude that the presence of straight lines provide strong evidence for our assumption that the distribution of gradients is in fact close to normally distributed.

## 5.3 Algorithm Comparisons

We compared our algorithm to the stochastic gradient method for $L_1$-regularized Log-linear models in [14], adapted for the $L_1$-regularized methods above. We use the following decay schedule for all curves over time labeled "SGD": $\eta = \eta_0 \frac{1}{t_0+t}$. In addition to this schedule, we also tested against SGD using the regret-minimizing schedule of [6] on the INRIA dataset: $\eta = \eta_0 \frac{1}{\sqrt{t_0+t}}$. After spending a significant amount of time hand-optimizing the hyper-parameters $\eta_0, t_0$, we found that settings $\eta_0 \approx 1$ for both rate schedules, and $t_0 \approx N/10$ (standard SGD) and $t_0 \approx (N/10)^2$ (regret-minimzing SGD) have worked well on our datasets. We ran all our algorithms – Lasso, Logistic Regression and SVM – with a value of $\epsilon = 0.05$ for both INRIA and VOC datasets.

Figures 2 show a comparison between our method and stochastic gradient descent on the INRIA and VOC datasets. Our method including the shrinking strategy is faster for the SVM, while methods without a data shrinking strategy, such as logistic regression, are still competitive (see Figure 2). In comparing our methods to the coordinate descent upon which ours are based, we see that our framework provides a considerable speedup over standard coordinate descent. We do this with a method which eventually uses the entire batch of data, so the tricks that enable SGD to converge in an $L_1$-regularized problem are not necessary. In terms of performance, our models are equivalent or near to published state of the art results for linear models [13, 15].

We also performed a comparison against SGD with an exponential decay schedule $\eta = \eta_0 e^{-\lambda t}$ on the Lasso problem (see Fig 3). Exponential decay schedules are known to work well in practice [14], but do not give the theoretical convergence guarantees of other schedules. For a range of values for $\eta_0$ and $\lambda$, we compare SGD against our algorithm with $\epsilon \in \{0.05, 0.2, 0.4\}$. From these experiments we conclude that changing $\epsilon$ from its standard value $0.05$ all the way to $0.4$ (recall that $\epsilon < 0.5$) has very little effect on accuracy and speed. This in contrast to SGD which required hyper-parameter tuning to achieve comparable performance.

To further demonstrate the robustness of our method to $\epsilon$, we performed 5 trials of logistic regression on the INRIA dataset with a wide range of values of $\epsilon$, with random initializations, shown in Figure 1. All choices of $\epsilon$ give a reasonable average precision, and the algorithm begins to become significantly slower only with $\epsilon > .3$.

## 6 Conclusions

We have introduced a new framework for optimization problems from a statistical, frequentist point of view. Every phase of the learning process has its own optimal batchsize. That is to say, we need only few data-cases early on in learning but many data-cases close to convergence. In fact, we argue that when we are using all of our data and cannot determine with statistical confidence that our update is in the correct direction, we should stop learning to avoid overfitting. These ideas are absent in the usual frequentist (a.k.a. maximum likelihood) and learning-theory approaches which formulate learning as the optimization of some loss function. A meaningful smallest length scale based on statistical considerations *is* present in Bayesian analysis through the notion of a posterior distribution. However, the most common inference technique in that domain, MCMC sampling, does not make use of the fact that less precision is needed during the first phases of learning (a.k.a. "burn-in") because any accept/reject rule requires all data-cases to be seen. Hence, our approach can be thought of as a middle ground that borrows from both learning philosophies.

Our approach also leverages the fact that some features are more predictive than others, and may deserve more attention during optimization. By predicting when updates will pass their statistical tests, we can update each feature approximately with the correct frequency.

The proposed algorithms feature a single variable that needs to be set. However, the variable has a clear meaning – the allowed probability that an update moves in the wrong direction. We have used $\epsilon = 0.05$ in all our experiments to showcase the robustness of the method.

Our method is not limited to $L_1$ methods or linear models; our framework can be used on any algorithm in which we take updates which are simple functions on averages over the data.

Relative to vanilla coordinate descent, our algorithms can handle dense datasets with $N >> p$. Relative to SGD[2] our method can be thought of as "self-annealing" in the sense that it increases its precision by adaptively increasing the dataset size. The advantages over SGD are therefore that we avoid tuning hyper-parameters of an annealing schedule and that we have an automated stopping criterion.

## Footnotes

[1]In practice we cap $k_{skip}$ at some maximum number of iterations (say 40).

[2]Recent benchmarks [16] show that a properly tuned SGD solver is highly competitive for large-scale problems [17].

# References

[1] L. Bottou and O. Bousquet. Learning using large datasets. In *Mining Massive DataSets for Security, NATO ASI Workshop Series. IOS Press, Amsterdam*, 2008.

[2] L. Bottou and O. Bousquet. The tradeoffs of large scale learning. *Advances in neural information processing systems*, 20:161–168, 2008.

[3] B. Yu. Embracing statistical challenges in the information technology age. *Technometrics, American Statistical Association and the American Society for Quality*, 49:237–248, 2007.

[4] N. Cesa-Bianchi, A. Conconi, and C. Gentile. On the generalization ability of on-line learning algorithms. *Information Theory, IEEE Transactions on*, 50(9):2050–2057, 2004.

[5] S. Shalev-Shwartz and N. Srebro. SVM optimization: inverse dependence on training set size. In *Proceedings of the 25th international conference on Machine learning*, pages 928–935. ACM, 2008.

[6] M. Zinkevich. Online convex programming and generalized infinitesimal gradient ascent. *Twentieth International Conference on Machine Learning*, 2003.

[7] P.L. Bartlett, E. Hazan, and A. Rakhlin. Adaptive online gradient descent. *Advances in Neural Information Processing Systems*, 21, 2007.

[8] A. Korattikara, L. Boyles, M. Welling, J. Kim, and H. Park. Statistical optimization of non-negative matrix factorization. AISTATS, 2011.

[9] J. Friedman, T. Hastie, H. Höfling, and R. Tibshirani. Pathwise coordinate optimization. *Annals of Applied Statistics*, 1(2):302–332, 2007.

[10] R.E. Fan, K.W. Chang, C.J. Hsieh, X.R. Wang, and C.J. Lin. LIBLINEAR: A library for large linear classification. *The Journal of Machine Learning Research*, 9:1871–1874, 2008.

[11] N. Le Roux, P.A. Manzagol, and Y. Bengio. Topmoumoute online natural gradient algorithm. In *Neural Information Processing Systems (NIPS)*. Citeseer, 2007.

[12] N. Dalal and B. Triggs. Histograms of oriented gradients for human detection. In *IEEE Computer Society Conference on Computer Vision and Pattern Recognition*, volume 1, page 886. Citeseer, 2005.

[13] M. Everingham, L. Van Gool, C. K. I. Williams, J. Winn, and A. Zisserman. The PASCAL Visual Object Classes Challenge 2007 (VOC2007) Results. http://www.pascal-network.org/challenges/VOC/voc2007/workshop/index.html.

[14] Yoshimasa Tsuruoka, Jun'ichi Tsujii, and Sophia Ananiadou. Stochastic gradient descent training for l1-regularized log-linear models with cumulative penalty. In *Proceedings of the Joint Conference of the 47th Annual Meeting of the ACL and the 4th International Joint Conference on Natural Language Processing of the AFNLP*, pages 477–485, Suntec, Singapore, August 2009. Association for Computational Linguistics.

[15] Navneet Dalal. *Finding People in Images and Video*. PhD thesis, Institut National Polytechnique de Grenoble / INRIA Grenoble, July 2006.

[16] Pascal large scale learning challenge. http://largescale.ml.tu-berlin.de/workshop/, 2008.

[17] A. Bordes, L. Bottou, and P. Gallinari. SGD-QN: Careful Quasi-Newton Stochastic Gradient Descent. *Journal of Machine Learning Research*, 10:1737–1754, 2009.

